# Regularized Distance Metric Learning: Theory and Algorithm

**Rong Jin**[1]     **Shijun Wang**[2]     **Yang Zhou**[1]

[1]Dept. of Computer Science & Engineering, Michigan State University, East Lansing, MI 48824

[2]Radiology and Imaging Sciences, National Institutes of Health, Bethesda, MD 20892

`rongjin@cse.msu.edu`  `wangshi@cc.nih.gov`  `zhouyang@msu.edu`

## Abstract

In this paper, we examine the generalization error of regularized distance metric learning. We show that with appropriate constraints, the generalization error of regularized distance metric learning could be independent from the dimensionality, making it suitable for handling high dimensional data. In addition, we present an efficient online learning algorithm for regularized distance metric learning. Our empirical studies with data classification and face recognition show that the proposed algorithm is (i) effective for distance metric learning when compared to the state-of-the-art methods, and (ii) efficient and robust for high dimensional data.

## 1   Introduction

Distance metric learning is a fundamental problem in machine learning and pattern recognition. It is critical to many real-world applications, such as information retrieval, classification, and clustering [6, 7]. Numerous algorithms have been proposed and examined for distance metric learning. They are usually classified into two categories: unsupervised metric learning and supervised metric learning. Unsupervised distance metric learning, or sometimes referred to as manifold learning, aims to learn a underlying low-dimensional manifold where the distance between most pairs of data points are preserved. Example algorithms in this category include ISOMAP [13] and Local Linear Embedding (LLE) [8]. Supervised metric learning attempts to learn distance metrics from side information such as labeled instances and pairwise constraints. It searches for the optimal distance metric that (a) keeps data points of the same classes close, and (b) keeps data points from different classes far apart. Example algorithms in this category include [17, 10, 15, 5, 14, 19, 4, 12, 16]. In this work, we focus on supervised distance metric learning.

Although a large number of studies were devoted to supervised distance metric learning (see the survey in [18] and references therein), few studies address the generalization error of distance metric learning. In this paper, we examine the generalization error for regularized distance metric learning. Following the idea of stability analysis [1], we show that with appropriate constraints, the generalization error of regularized distance metric learning is independent from the dimensionality of data, making it suitable for handling high dimensional data. In addition, we present an online learning algorithm for regularized distance metric learning, and show its regret bound. Note that although online metric learning was studied in [9], our approach is advantageous in that (a) it is computationally more efficient in handling the constraint of SDP cone, and (b) it has a proved regret bound while [9] only shows a mistake bound for the datasets that can be separated by a Mahalanobis distance. To verify the efficacy and efficiency of the proposed algorithm for regularized distance metric learning, we conduct experiments with data classification and face recognition. Our empirical results show that the proposed online algorithm is (1) effective for metric learning compared to the state-of-the-art methods, and (2) robust and efficient for high dimensional data.

## 2 Regularized Distance Metric Learning

Let $\mathcal{D} = \{z_i = (x_i, y_i), i = 1, \ldots, n\}$ denote the labeled examples, where $x_k = (x_k^1, \ldots, x_k^d) \in \mathbb{R}^d$ is a vector of $d$ dimension and $y_i \in \{1, 2, \ldots, m\}$ is class label. In our study, we assume that the norm of any example is upper bounded by $R$, i.e., $\sup_x |x|_2 \leq R$. Let $A \in \mathbf{S}_+^{d \times d}$ be the distance metric to be learned, where the distance between two data points $x$ and $x'$ is calculated as $|x - x'|_A^2 = (x - x')^\top A(x - x')$.

Following the idea of maximum margin classifiers, we have the following framework for regularized distance metric learning:

$$\min_A \left\{ \frac{1}{2}|A|_F^2 + \frac{2C}{n(n-1)} \sum_{i<j} g\left(y_{i,j}\left[1 - |x_i - x_j|_A^2\right]\right) : A \succeq 0, \operatorname{tr}(A) \leq \eta(d) \right\} \tag{1}$$

where

- $y_{i,j}$ is derived from class labels $y_i$ and $y_j$, i.e., $y_{i,j} = 1$ if $y_i = y_j$ and $-1$ otherwise.
- $g(z)$ is the loss function. It outputs a small value when $z$ is a large positive value, and a large value when $z$ is large negative. We assume $g(z)$ to be convex and Lipschitz continuous with Lipschitz constant $L$.
- $|A|_F^2$ is the regularizer that measures the complexity of the distance metric $A$.
- $\operatorname{tr}(A) \leq \eta(d)$ is introduced to ensure a bounded domain for $A$. As will be revealed later, this constraint will become active only when the constraint constant $\eta(d)$ is sublinear in $d$, i.e., $\eta \sim O(d^p)$ with $p < 1$. We will also show how this constraint could affect the generalization error of distance metric learning.

## 3 Generalization Error

Let $A_{\mathcal{D}}$ be the distance metric learned by the algorithm in (1) from the training examples $\mathcal{D}$. Let $I_{\mathcal{D}}(A)$ denote the empirical loss , i.e.,

$$I_{\mathcal{D}}(A) = \frac{2}{n(n-1)} \sum_{i<j} g\left(y_{i,j}\left[1 - |x_i - x_j|_A^2\right]\right) \tag{2}$$

For the convenience of presentation, we also write $g\left(y_{i,j}(1 - |x_i - x_j|_A^2)\right) = V(A, z_i, z_j)$ to highlight its dependence on $A$ and two examples $z_i$ and $z_j$. We denote by $I(A)$ the loss of $A$ over the true distribution, i.e.,

$$I(A) = \mathbb{E}_{(z_i, z_j)}[V(A, z_i, z_j)] \tag{3}$$

Given the empirical loss $I_{\mathcal{D}}(A)$ and the loss over the true distribution $I(A)$, we define the estimation error as

$$D_{\mathcal{D}} = I(A_{\mathcal{D}}) - I_{\mathcal{D}}(A_{\mathcal{D}}) \tag{4}$$

In order to show the behavior of estimation error, we follow the analysis based on the stability of the algorithm [1]. The uniform stability of an algorithm determines the stability of the algorithm when one of the training examples is replaced with another. More specifically, an algorithm $\mathcal{A}$ has uniform stability $\beta$ if

$$\forall(\mathcal{D}, z), \ \forall i, \ \sup_{u,v} |V(A_{\mathcal{D}}, u, v) - V(A_{\mathcal{D}^{z,i}}, u, v)| \leq \beta \tag{5}$$

where $\mathcal{D}^{z,i}$ stands for the new training set that is obtained by replacing $z_j \in \mathcal{D}$ with a new example $z$. We further define $\beta = \kappa/n$ as the uniform stability $\beta$ behaves like $O(1/n)$.

The advantage of using stability analysis for the generalization error of regularized distance metric learning. This is because the example pair $(z_i, z_j)$ used for training distance metrics are not I.I.D. although $z_i$ is, making it difficult to directly utilize the results from statistical learning theory.

In the analysis below, we first show how to derive the generalization error bound for regularized distance metric learning given the uniform stability $\beta$ (or $\kappa$). We then derive the uniform stability constant for the regularized distance metric learning framework in (1).

## 3.1 Generalization Error Bound for Given Uniform Stability

Analysis in this section follows closely [1], and we therefore omit the detailed proofs.

Our analysis utilizes the McDiarmid inequality that is stated as follows.

**Theorem 1.** *(McDiarmid Inequality) Given random variables $\{v_i\}_{i=1}^l, v_i'$, and a function $F: v^l \to \mathbb{R}$ satisfying*

$$\sup_{v_1,\ldots,v_l,v_i'} \left| F(v_1,\ldots,v_l) - F(v_1,\ldots,v_{i-1},v_i',v_{i+1},\ldots,v_l) \right| \leq c_i,$$

*the following statement holds*

$$\Pr\left( |F(v_1,\ldots,v_l) - \mathbb{E}(F(v_1,\ldots,v_l))| > \epsilon \right) \leq 2\exp\left( -\frac{2\epsilon}{\sum_{i=1}^l c_i^2} \right)$$

To use the McDiarmid inequality, we first compute $\mathbb{E}(D_{\mathcal{D}})$.

**Lemma 1.** *Given a distance metric learning algorithm $\mathcal{A}$ has uniform stability $\kappa/n$, we have the following inequality for $\mathbb{E}(D_{\mathcal{D}})$*

$$\mathbb{E}(D_{\mathcal{D}}) \leq 2\frac{\kappa}{n} \tag{6}$$

*where $n$ is the number of training examples in $\mathcal{D}$.*

The result in the following lemma shows that the condition in McDiarmid inequality holds.

**Lemma 2.** *Let $\mathcal{D}$ be a collection of $n$ randomly selected training examples, and $\mathcal{D}^{i,z}$ be the collection of examples that replaces $z_i$ in $\mathcal{D}$ with example $z$. We have $|D_{\mathcal{D}} - D_{\mathcal{D}^{i,z}}|$ bounded as follows*

$$|D_{\mathcal{D}} - D_{\mathcal{D}^{i,z}}| \leq \frac{2\kappa + 8L\eta(d) + 2g_0}{n} \tag{7}$$

*where $g_0 = \sup_{z,z'} |V(0,z,z')|$ measures the largest loss when distance metric $A$ is $0$.*

Combining the results in Lemma 1 and 2, we can now derive the the bound for the generalization error by using the McDiarmid inequality.

**Theorem 2.** *Let $\mathcal{D}$ denote a collection of $n$ randomly selected training examples, and $A_{\mathcal{D}}$ be the distance metric learned by the algorithm in (1) whose uniform stability is $\kappa/n$. With probability $1 - \delta$, we have the following bound for $I(A_{\mathcal{D}})$*

$$I(A_{\mathcal{D}}) - I_{\mathcal{D}}(A_{\mathcal{D}}) \leq \frac{2\kappa}{n} + (2\kappa + 4L\eta(d) + 2g_0)\sqrt{\frac{\ln(2/\delta)}{2n}} \tag{8}$$

## 3.2 Generalization Error for Regularized Distance Metric Learning

First, we show that the superium of $\operatorname{tr}(A_{\mathcal{D}})$ is $O(d^{1/2})$, which verifies that $\eta(d)$ should behave sublinear in $d$. This is summarized by the following proposition.

**Proposition 1.** *The trace constraint in (1) will be activated only when*

$$\eta(d) \leq \sqrt{2dg_0C} \tag{9}$$

*where $g_0 = \sup_{z,z'} |V(\mathbf{0}, z, z')|$.*

*Proof.* It follows directly from $[\operatorname{tr}(A_{\mathcal{D}})/d]^2 \leq |A_{\mathcal{D}}|_F^2 \leq 2C \sup_{z,z'} |V(\mathbf{0}, z, z')| \leq Cg_0$. □

To bound the uniform stability, we need the following proposition

**Proposition 2.** *For any two distance metrics $A$ and $A'$, we have the following inequality hold for any examples $z_u$ and $z_v$*

$$|V(A, z_u, z_v) - V(A', z_u, z_v)| \leq 4LR^2|A - A'|_F \tag{10}$$

The above proposition follows directly from the fact that (a) $V(A, z, z')$ is Lipschitz continuous and (b) $|x|_2 \leq R$ for any example $x$. The following lemma bounds $|A_\mathcal{D} - A_{\mathcal{D}'}|_F$.

**Lemma 3.** *Let $\mathcal{D}$ denote a collection of $n$ randomly selected training examples, and by $z = (x, y)$ a randomly selected example. Let $A_\mathcal{D}$ be the distance metric learned by the algorithm in (1). We have*

$$|A_\mathcal{D} - A_{\mathcal{D}^{i,z}}|_F \leq \frac{8CLR^2}{n} \tag{11}$$

The proof of the above lemma can be found in Appendix A.

By putting the results in Lemma 3 and Proposition 2, we have the following theorem for the stability of the Frobenius norm based regularizer.

**Theorem 3.** *The uniform stability for the algorithm in (1) using the Frobenius norm regularizer, denoted by $\beta$, is bounded as follows*

$$\beta = \frac{\kappa}{n} \leq \frac{32CL^2R^4}{n} \tag{12}$$

*where $\kappa = 32CL^2R^4$*

Combing Theorem 3 and 2, we have the following theorem for the generalization error of distance metric learning algorithm in (1) using the Frobenius norm regularizer

**Theorem 4.** *Let $\mathcal{D}$ be a collection of n randomly selected examples, and $A_\mathcal{D}$ be the distance metric learned by the algorithm in (1) with $h(A) = |A|_F^2$. With probability $1 - \delta$, we have the following bound for the true loss function $I(A_\mathcal{D})$ where $A_\mathcal{D}$ is learned from (1) using the Frobenius norm regularizer*

$$I(A_\mathcal{D}) - I_\mathcal{D}(A_\mathcal{D}) \leq \frac{32CL^2R^4}{n} + \left(32CL^2R^4 + 4Ls(d) + 2g_0\right)\sqrt{\frac{\ln(2/\delta)}{2n}} \tag{13}$$

*where $s(d) = \min\left(\sqrt{2dg_0C}, \eta(d)\right)$.*

**Remark**  The most important feature in the estimation error is that it converges in the order of $O(s(d)/\sqrt{n})$. By choosing $\eta(d)$ to have a low dependence of $d$ (i.e., $\eta(d) \sim d^p$ with $p \ll 1$), the proposed framework for regularized distance metric learning will be robust to the high dimensional data. In the extreme case, by setting $\eta(d)$ to be a constant, the estimation error will be independent from the dimensionality of data.

## 4  Algorithm

In this section, we discuss an efficient algorithm for solving (1). We assume a hinge loss for $g(z)$, i.e., $g(z) = \max(0, b - z)$, where $b$ is the classification margin. To design an online learning algorithm for regularized distance metric learning, we follow the theory of gradient based online learning [2] by defining potential function $\Phi(A) = |A|_F^2/2$. Algorithm 1 shows the online learning algorithm.

The theorem below shows the regret bound for the online learning algorithm in Figure 1.

**Theorem 5.** *Let the online learning algorithm 1 run with learning rate $\lambda > 0$ on a sequence $(x_t, x_t'), y_t, t = 1, \dots, n$. Assume $|x|_2 \leq R$ for all the training examples. Then, for all distance metric $M \in S_+^{d \times d}$, we have*

$$\widehat{L}_n \leq \frac{1}{1 - 8R^4\lambda/b}\left(L_n(M) + \frac{1}{2\lambda}|M|_F^2\right)$$

*where*

$$Ł_n(M) = \sum_{t=1}^n \max\left(0, b - y_t(1 - |x_t - x_t'|_M^2)\right), \widehat{L}_n = \sum_{t=1}^n \max\left(0, b - y_t(1 - |x_t - x_t'|_{A_{t-1}}^2)\right)$$

**Algorithm 1** Online Learning Algorithm for Regularized Distance Metric Learning

1: INPUT: predefined learning rate $\lambda$
2: Initialize $A_0 = \mathbf{0}$
3: **for** $t = 1, \ldots, T$ **do**
4:     Receive a pair of training examples $\{(x_t^1, y_t^1), (x_t^2, y_t^2)\}$
5:     Compute the class label $y_t$: $y_t = +1$ if $y_t^1 = y_t^2$, and $y_t = -1$ otherwise.
6:     **if** the training pair $(x_t^1, x_t^2)$, $y_t$ is classified correctly, i.e., $y_t \left( 1 - |x_t^1 - x_t^2|_{A_{t-1}}^2 \right) > 0$ **then**
7:         $A_t = A_{t-1}$.
8:     **else**
9:         $A_t = \pi_{S_+}(A_{t-1} - \lambda y_t(x_t - x_t')(x_t - x_t')^\top)$, where $\pi_{S_+}(M)$ projects matrix $M$ into the SDP cone.
10:     **end if**
11: **end for**

The proof of this theorem can be found in Appendix B. Note that the above online learning algorithm require computing $\pi_{S_+}(M)$, i.e., projecting matrix $M$ onto the SDP cone, which is expensive for high dimensional data. To address this challenge, first notice that $M' = \pi_{S_+}(M)$ is equivalent to the optimization problem $M' = \arg\min_{M' \succeq 0} |M' - M|_F$. We thus approximate $A_t = \pi_{S_+}(A_{t-1} - \lambda y_t(x_t - x_t')(x_t - x_t')^\top)$ with $A_t = A_{t-1} - \lambda_t y_t(x_t - x_t')(x_t - x_t')^\top$ where $\lambda_t$ is computed as follows

$$\lambda_t = \arg\min_{\lambda_t} \left\{ |\lambda_t - \lambda| : \lambda_t \in [0, \lambda], A_{t-1} - \lambda_t y_t(x_t - x_t')(x_t - x_t')^\top \succeq 0 \right\} \tag{14}$$

The following theorem shows the solution to the above optimization problem.

**Theorem 6.** *The optimal solution $\lambda_t$ to the problem in (14) is expressed as*

$$\lambda_t = \begin{cases} \lambda & y_t = -1 \\ \min\left( \lambda, [(x_t - x_t')^\top A_{t-1}^{-1}(x_t - x_t')]^{-1} \right) & y_t = +1 \end{cases}$$

Proof of this theorem can be found in the supplementary materials. Finally, the quantity $(x_t - x_t')A_{t-1}^{-1}(x_t - x_t')$ can be computed by solving the following optimization problem

$$\max_u 2u^\top(x_t - x_t') - u^\top A u$$

whose optimal value can be computed efficiently using the conjugate gradient method [11].

Note that compared to the online metric learning algorithm in [9], the proposed online learning algorithm for metric learning is advantageous in that (i) it is computationally more efficient by avoiding projecting a matrix into a SDP cone, and (ii) it has a provable regret bound while [9] only presents the mistake bound for the separable datasets.

## 5 Experiments

We conducted an extensive study to verify both the efficiency and the efficacy of the proposed algorithms for metric learning. For the convenience of discussion, we refer to the propoesd online distance metric learning algorithm as **online-reg**. To examine the efficacy of the learned distance metric, we employed the $k$ Nearest Neighbor ($k$-NN) classifier. Our hypothesis is that the better the distance metric is, the higher the classification accuracy of $k$-NN will be. We set $k = 3$ for $k$-NN for all the experiments according to our experience.

We compare our algorithm to the following six state-of-the-art algorithms for distance metric learning as baselines: (1) **Euclidean** distance metric; (2) **Mahalanobis** distance metric, which is computed as the inverse of covariance matrix of training samples, i.e., $(\sum_{i=1}^n x_i x_i)^{-1}$; (3) **Xing**'s algorithm proposed in [17]; (4) **LMNN**, a distance metric learning algorithm based on the large margin nearest neighbor classifier [15]; (5) **ITML**, an Information-theoretic metric learning based on [4]; and (6) Relevance Component Analysis (**RCA**) [10]. We set the maximum number of iterations for Xing's method to be $10,000$. The number of target neighbors in LMNN and parameter $\gamma$ in ITML

Table 1: Classification error (%) of a $k$-NN ($k = 3$) classifier on the ten UCI data sets using seven different metrics. Standard deviation is included.

| Dataset | Eclidean | Mahala | Xing | LMNN | ITML | RCA | Online-reg |
|---|---|---|---|---|---|---|---|
| 1 | $19.5 \pm 2.2$ | $18.8 \pm 2.5$ | $29.3 \pm 17.2$ | $13.8 \pm 2.5$ | $8.6 \pm 1.7$ | $17.4 \pm 1.5$ | $13.2 \pm 2.2$ |
| 2 | $39.9 \pm 2.3$ | $6.7 \pm 0.6$ | $40.1 \pm 2.6$ | $3.6 \pm 1.1$ | $40.0 \pm 2.3$ | $3.8 \pm 0.4$ | $3.7 \pm 1.2$ |
| 3 | $36.0 \pm 2.0$ | $42.1 \pm 4.0$ | $43.5 \pm 12.5$ | $33.1 \pm 0.6$ | $39.8 \pm 3.3$ | $41.6 \pm 0.7$ | $37.3 \pm 4.1$ |
| 4 | $4.0 \pm 1.7$ | $10.4 \pm 2.7$ | $3.1 \pm 2.0$ | $3.9 \pm 1.6$ | $3.2 \pm 1.6$ | $2.9 \pm 1.5$ | $3.2 \pm 1.3$ |
| 5 | $30.6 \pm 1.9$ | $29.1 \pm 2.1$ | $30.6 \pm 1.9$ | $29.6 \pm 1.8$ | $28.8 \pm 2.1$ | $28.6 \pm 2.3$ | $27.7 \pm 1.3$ |
| 6 | $25.4 \pm 4.2$ | $18.4 \pm 3.4$ | $23.3 \pm 3.4$ | $15.2 \pm 3.1$ | $17.1 \pm 4.1$ | $13.9 \pm 2.2$ | $12.9 \pm 2.2$ |
| 7 | $31.9 \pm 2.8$ | $10.0 \pm 2.8$ | $24.6 \pm 7.5$ | $4.5 \pm 2.4$ | $28.7 \pm 3.7$ | $1.8 \pm 1.5$ | $1.8 \pm 1.1$ |
| 8 | $18.9 \pm 0.5$ | $37.3 \pm 0.5$ | $16.1 \pm 0.6$ | $18.4 \pm 0.4$ | $23.3 \pm 1.3$ | $30.6 \pm 0.7$ | $19.8 \pm 0.6$ |
| 9 | $2.0 \pm 0.4$ | $6.1 \pm 0.5$ | $12.4 \pm 0.8$ | $1.6 \pm 0.3$ | $2.5 \pm 0.4$ | $2.8 \pm 0.4$ | $2.9 \pm 0.4$ |

Table 2: $p$-values of the Wilcoxon signed-rank test of the 7 methods on the 9 datasets.

| Methods | Eclidean | Mahala | Xing | LMNN | ITML | RCA | Online-reg |
|---|---|---|---|---|---|---|---|
| Euclidean | 1.000 | 0.734 | 0.641 | 0.004 | 0.496 | 0.301 | 0.129 |
| Mahala | 0.734 | 1.000 | 0.301 | 0.008 | 0.570 | 0.004 | 0.004 |
| Xing | 0.641 | 0.301 | 1.000 | 0.027 | 0.359 | 0.074 | 0.027 |
| LMNN | 0.004 | 0.008 | 0.027 | 1.000 | 0.129 | 0.496 | 0.734 |
| ITML | 0.496 | 0.570 | 0.359 | 0.129 | 1.000 | 0.820 | 0.164 |
| RCA | 0.301 | 0.004 | 0.074 | 0.496 | 0.820 | 1.000 | 0.074 |
| Online-reg | 0.129 | 0.004 | 0.027 | 0.734 | 0.164 | 0.074 | 1.000 |

were tuned by cross validation over the range from $10^{-4}$ to $10^4$. All the algorithms are implemented and run using Matlab. All the experiment are run on a AMD Processor 2.8G machine, with 8GMB RAM and Linux operation system.

## 5.1 Experiment (I): Comparison to State-of-the-art Algorithms

We conducted experiments of data classification over the following nine datasets from UCI repository: (1) *balance-scale*, with 3 classes, 4 features, and 625 instances; (2) *breast-cancer*, with 2 classes, 10 features, and 683 instance; (3) *glass*, with 6 classes, 9 features, and 214 instances; (4) *iris*, with 3 classes, 4 features, and 150 instances; (5) *pima*, with 2 classes, 8 features, and 768 instances; (6) *segmentation*, with 7 classes, 19 features, and 210 instances; (7)*wine*, with 3 classes, 13 features, and 178 instances; (8) *waveform*, with 3 classes, 21 features, and 5000 instances; (9) *optdigits*, with 10 classes, 64 features, 3823 instances. For all the datasets, we randomly select $50\%$ samples for training, and use the remaining samples for testing. Table 1 shows the classification errors of all the metric learning methods over 9 datasets averaged over 10 runs, together with the standard deviation. We observe that the proposed metric learning algorithm deliver performance that comparable to the state-of-the-art methods. In particular, for almost all datasets, the classification accuracy of the proposed algorithm is close to that of LMNN, which has yielded overall the best performance among six baseline algorithms. This is consistent with the results of the other studies, which show LMNN is among the most effective algorithms for distance metric learning.

To further verify if the proposed method performs statistically better than the baseline methods, we conduct statistical test by using Wilcoxon signed-rank test [3]. The Wilcoxon signed-rank test is a non-parametric statistical hypothesis test for the comparisons of two related samples. It is known to be safer than the Student's t-test because it does not assume normal distributions. From table 2, we find that the regularized distance metric learning improves the classification accuracy significantly compared to Mahalanobis distance, Xing's method and RCA at significant level 0.1. It performs slightly better than ITML and is comparable to LMNN.

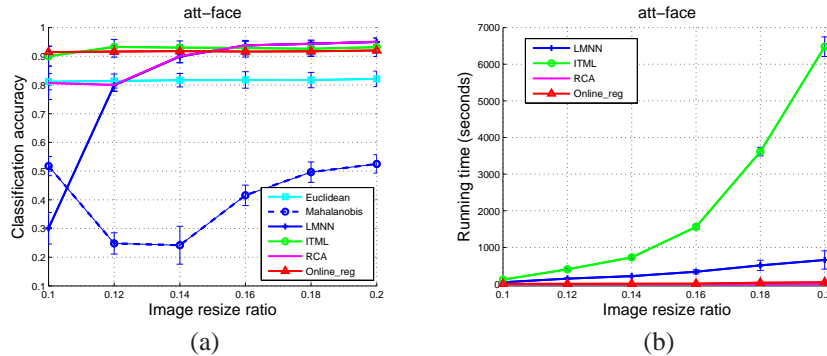

Figure 1: (a) Face recognition accuracy of $k$NN and (b) running time of LMNN, ITML, RCA and online_reg algorithms on the "att-face" dataset with varying image sizes.

## 5.2 Experiment (II): Results for High Dimensional Data

To evaluate the dependence of the regularized metric learning algorithms on data dimensions, we tested it by the task of face recognition. The AT&T face database [1] is used in our study. It consists of grey images of faces from $40$ distinct subjects, with ten pictures for each subject. For every subject, the images were taken at different times, with varied the lighting condition and different facial expressions (open/closed-eyes, smiling/not-smiling) and facial details (glasses/no-glasses). The original size of each image is $112 \times 92$ pixels, with $256$ grey levels per pixel.

To examine the sensitivity to data dimensionality, we vary the data dimension (i.e., the size of images) by compressing the original images into size different sizes with the image aspect ratio preserved. The image compression is achieved by bicubic interpolation (the output pixel value is a weighted average of pixels in the nearest 4-by-4 neighborhood). For each subject, we randomly spit its face images into training set and test set with ratio $4 : 6$. A distance metric is learned from the collection of training face images, and is used by the $k$NN classifier ($k = 3$) to predict the subject ID of the test images. We conduct each experiment $10$ times, and report the classification accuracy by averaging over $40$ subjects and $10$ runs. Figure 1 (a) shows the average classification accuracy of the $k$NN classifier using different distance metric learning algorithms. The running times of different metric learning algorithms for the same dataset is shown in Figure 1 (b). Note that we exclude Xing's method in comparison because its extremely long computational time. We observed that with increasing image size (dimensions), the regularized distance metric learning algorithm yields stable performance, indicating that the it is resilient to high dimensional data. In contrast, for almost all the baseline methods except ITML, their performance varied significantly as the size of the input image changed. Although ITML yields stable performance with respect to different size of images, its high computational cost (Figure 1), arising from solving a Bregman optimization problem in each iteration, makes it unsuitable for high-dimensional data.

## 6 Conclusion

In this paper, we analyze the generalization error of regularized distance metric learning. We show that with appropriate constraint, the regularized distance metric learning could be robust to high dimensional data. We also present efficient learning algorithms for solving the related optimization problems. Empirical studies with face recognition and data classification show the proposed approach is (i) robust and efficient for high dimensional data, and (ii) comparable to the state-of-the-art approaches for distance learning. In the future, we plan to investigate different regularizers and their effect for distance metric learning.

**ACKNOWLEDGEMENTS**

The work was supported in part by the National Science Foundation (IIS-0643494) and the U. S. Army Research Laboratory and the U. S. Army Research Office (W911NF-09-1-0421). Any opinions, findings, and conclusions or recommendations expressed in this material are those of the authors and do not necessarily reflect the views of NSF and ARO.

## Appendix A: Proof of Lemma 3

*Proof.* We introduce the Bregmen divergence for the proof of this lemma. Given a convex function of matrix $\varphi(X)$, the Bregmen divergence between two matrices $A$ and $B$ is computed as follows:

$$d_\varphi(A, B) = \varphi(B) - \varphi(A) - \text{tr}\left(\nabla\varphi(A)^\top (B - A)\right)$$

We define convex function $N(X)$ and $V_\mathcal{D}(X)$ as follows:

$$N(X) = \|X\|_F^2, \quad V_\mathcal{D}(X) = \frac{2}{n(n-1)} \sum_{i<j} V(X, z_i, z_j)$$

and furthermore convex function $T_\mathcal{D}(X) = N(X) + CV_\mathcal{D}(X)$. We thus have

$$d_N(A_\mathcal{D}, A_{\mathcal{D}^{i,z}}) + d_N(A_{\mathcal{D}^{i,z}}, A_\mathcal{D}) \leq d_{T_\mathcal{D}}(A_\mathcal{D}, A_{\mathcal{D}^{i,z}}) + d_{T_{\mathcal{D}^{i,z}}}(A_{\mathcal{D}^{i,z}}, A_\mathcal{D})$$

$$= \frac{C}{n(n-1)} \sum_{j \neq i} [V(A_{\mathcal{D}^{i,z}}, z_i, z_j) - V(A_{\mathcal{D}^{i,z}}, z, z_j) + V(A_\mathcal{D}, z, z_j) - V(A_\mathcal{D}, z_i, z_j)]$$

$$\leq \frac{8CLR^2}{n} |A_\mathcal{D} - A_{\mathcal{D}^{i,z}}|_F$$

The first inequality follows from the fact that both $N(X)$ and $V_\mathcal{D}(X)$ are convex in $X$. The second step holds because matrix $A_\mathcal{D}$ and $A_{\mathcal{D}^{i,z}}$ minimize the objective function $T_\mathcal{D}(X)$ and $T_{\mathcal{D}^{i,z}}(X)$, respectively, and therefore

$$(A_{\mathcal{D}^{i,z}} - A_\mathcal{D})^\top \nabla T_\mathcal{D}(A_\mathcal{D}) \geq 0, \quad (A_\mathcal{D} - A_{\mathcal{D}^{i,z}})^\top \nabla T_{\mathcal{D}^{i,z}}(A_{\mathcal{D}^{i,z}}) \geq 0$$

Since $d_N(A, B) = \|A - B\|_F^2$, we therefore have

$$|A_\mathcal{D} - A_{\mathcal{D}^{i,z}}|_F^2 \leq \frac{8CLR^2}{n} |A_\mathcal{D} - A_{\mathcal{D}^{i,z}}|_F,$$

which leads to the result in the lemma. $\square$

## Appendix B: Proof of Theorem 7

*Proof.* We denote by $A'_t = A_{t-1} - \lambda y(x_t - x'_t)(x_t - x'_t)^\top$ and $A_t = \pi_{S_+}(A'_t)$. Following Theorem 11.1 and Theorem 11.4 [2], we have

$$\widehat{L}_n - L_n(M) \leq \frac{1}{\lambda} D_{\Phi^*}(M, A_0) + \frac{1}{\lambda} \sum_{t=1}^n D_{\Phi^*}(A_{t-1}, A'_t)$$

where

$$D_{\Phi^*}(A, B) = \frac{1}{2} |A - B|_F^2, \Phi(A) = \Phi^*(A) = \frac{1}{2} |A|_F^2$$

Using the relation $A'_t = A_{t-1} - \lambda y(x_t - x'_t)(x_t - x'_t)^\top$ and $A_0 = \mathbf{0}$, we have

$$\widehat{L}_n - L_n(M) \leq \frac{1}{2\lambda} |M|_F^2 + \frac{1}{2\lambda} \sum_{t=1}^n I\left[y_t(1 - |x_t - x'_t|_{A_{t-1}}^2) < 0\right] |x_t - x'_t|^4$$

By assuming $|x|_2 \leq R$ for any training example, we have $|x_t - x'_t|_2^4 \leq 16R^4$. Since

$$\sum_{t=1}^n I\left[y_t(1 - |x_t - x'_t|_{A_{t-1}}^2) < 0\right] |x_t - x'_t|^4 \leq \sum_{t=1}^n \max(0, b - y_t(1 - |x_t - x'_t|_{A_{t-1}}^2)) \frac{16R^4}{b} = \frac{16R^4}{b} \widehat{L}_n$$

we thus have the result in the theorem $\square$

## Footnotes

[1] http://www.cl.cam.ac.uk/research/dtg/attarchive/facedatabase.html

# References

[1] Bousquet, Olivier, and André Elisseeff. Stability and generalization. *Journal of Machine Learning Research*, 2:499–526, March 2002.

[2] Nicolo Cesa-Bianchi and Gabor Lugosi. *Prediction, Learning, and Games*. Cambridge University Press, New York, NY, USA, 2006.

[3] G.W. Corder and D.I. Foreman. *Nonparametric Statistics for Non-Statisticians: A Step-by-Step Approach*. New Jersey: Wiley, 2009.

[4] J.V. Davis, B. Kulis, P. Jain, S. Sra, and I.S. Dhillon. Information-theoretic metric learning. In *Proceedings of the 24th international conference on Machine Learning*, 2007.

[5] A. Globerson and S. Roweis. Metric learning by collapsing classes. In *Advances in Neural Information Processing Systems*, 2005.

[6] Steven C.H. Hoi, Wei Liu, and Shih-Fu Chang. Semi-supervised distance metric learning for collaborative image retrieval. In *Proceedings of IEEE Conference on Computer Vision and Pattern Recognition (CVPR)*, 2008.

[7] Steven C.H. Hoi, Wei Liu, Michael R. Lyu, and Wei-Ying Ma. Learning distance metrics with contextual constraints for image retrieval. In *Proceedings of IEEE Conference on Computer Vision and Pattern Recognition (CVPR)*, 2006.

[8] L. K. Saul and S. T. Roweis. Think globally, fit locally: Unsupervised learning of low dimensional manifolds. *Journal of Machine Learning Research*, 4, 2003.

[9] Shai Shalev-Shwartz, Yoram Singer, and Andrew Y. Ng. Online and batch learning of pseudo-metrics. In *Proceedings of the twenty-first international conference on Machine learning*, pages 94–101, 2004.

[10] N. Shental, T. Hertz, D. Weinshall, and M. Pavel. Adjustment learning and relevant component analysis. In *Proceedings of the Seventh European Conference on Computer Vision*, volume 4, pages 776–792, 2002.

[11] Jonathan R Shewchuk. An introduction to the conjugate gradient method without the agonizing pain. Technical report, Carnegie Mellon University, Pittsburgh, PA, USA, 1994.

[12] Luo Si, Rong Jin, Steven C. H. Hoi, and Michael R. Lyu. Collaborative image retrieval via regularized metric learning. In *ACM Multimedia Systems Journal (MMSJ)*, 2006.

[13] J.B. Tenenbaum, V. de Silva, and J. C. Langford. A global geometric framework for nonlinear dimensionality reduction. *Science*, 290, 2000.

[14] I.W. Tsang, P.M. Cheung, and J.T. Kwok. Kernel relevant component analysis for distance metric learning. In *IEEE International Joint Conference on Neural Networks (IJCNN)*, 2005.

[15] K. Weinberger, J. Blitzer, and L. Saul. Distance metric learning for large margin nearest neighbor classification. In *Advances in Neural Information Processing Systems*, 2005.

[16] Lei Wu, Steven C.H. Hoi, Rong Jin, Jianke Zhu, and Nenghai Yu. Distance metric learning from uncertain side information with application to automated photo tagging. In *Proceedings of ACM International Conference on Multimedia (MM)*, 2009.

[17] E.P. Xing, A.Y. Ng, M.I. Jordan, and S. Russell. Distance metric learning, with application to clustering with side-information. In *Advances in Neural Information Processing Systems*, 2002.

[18] L. Yang and R. Jin. Distance metric learning: A comprehensive survey. *Michigan State University, Tech. Rep.*, 2006.

[19] L. Yang, R. Jin, R. Sukthankar, and Y. Liu. An efficient algorithm for local distance metric learning. In *the Proceedings of the Twenty-First National Conference on Artificial Intelligence Proceedings (AAAI)*, 2006.

